# Semiparametric Approach to Multichannel Blind Deconvolution of Nonminimum Phase Systems

**L.-Q. Zhang, S. Amari and A. Cichocki**
Brain-style Information Systems Research Group, BSI
The Institute of Physical and Chemical Research
Wako shi, Saitama 351-0198, JAPAN
*zha@open.brain.riken.go.jp*
*{amari,cia}@brain.riken.go.jp*

## Abstract

In this paper we discuss the semiparametric statistical model for blind deconvolution. First we introduce a Lie Group to the manifold of non-causal FIR filters. Then blind deconvolution problem is formulated in the framework of a semiparametric model, and a family of estimating functions is derived for blind deconvolution. A natural gradient learning algorithm is developed for training noncausal filters. Stability of the natural gradient algorithm is also analyzed in this framework.

## 1 Introduction

Recently blind separation/deconvolution has been recognized as an increasing important research area due to its rapidly growing applications in various fields, such as telecommunication systems, image enhancement and biomedical signal processing. Refer to review papers [7] and [13] for details. A semiparametric statistical model treats a family of probability distributions specified by a finite-dimensional parameter of interest and an infinite-dimensional nuisance parameter [12]. Amari and Kumon [10] have proposed an approach to semiparametric statistical models in terms of estimating functions and elucidated their geometric structures and efficiencies by information geometry [1]. Blind source separation can be formulated in the framework of semiparametric statistical models. Amari and Cardoso [5] applied information geometry of estimating functions to blind source separation and derived an admissible class of estimating functions which includes efficient estimators. They showed that the manifold of mixtures is $m-$curvature free, so that we can design algorithms of blind separation without taking much care of misspecification of source probability functions.

The theory of estimating functions has also been applied to the case of instantaneous mixtures, where independent source signals have unknown temporal correlations [3]. It is also applied to derive efficiency and superefficiency of demixing learning algorithms [4].

Most of these theories treat only blind source separation of instantaneous mixtures. It is only recently that the natural gradient approach has been proposed for multichannel blind

deconvolution [8], [18]. The present paper extends the geometrical theory of estimating functions to the semiparametric model of multichannel blind deconvolution. For the limited space, the detailed derivations and proofs are left to a full paper.

## 2  Blind Deconvolution Problem

In this paper, as a convolutive mixing model, we consider a multichannel linear time-invariant (LTI) systems, with no poles on the unit circle, of the form

$$\mathbf{x}(k) = \sum_{p=-\infty}^{\infty} \mathbf{H}_p \mathbf{s}(k-p),\qquad(1)$$

where $\mathbf{s}(k)$ is an $n$-dimensional vector of source signals which are spatially mutually independent and temporarily identically independently distributed, and $\mathbf{x}(k)$ is an $n-$dimensional sensor vector at time $k$, $k = 1, 2, \cdots$. We denote the unknown mixing filter by $\mathbf{H}(z) = \sum_{p=-\infty}^{\infty} \mathbf{H}_p z^{-p}$. The goal of multichannel blind deconvolution is to retrieve source signals $\mathbf{s}(k)$ only using sensor signals $\mathbf{x}(k), k = 1, 2, \cdots$, and certain knowledge of the source signal distributions and statistics. We carry out blind deconvolution by using another multichannel LTI system of the form

$$\mathbf{y}(k) = \mathbf{W}(z)\mathbf{x}(k),\qquad(2)$$

where $\mathbf{W}(z) = \sum_{p=-N}^{N} \mathbf{W}_p z^{-p}$, $N$ is the length of FIR filter $\mathbf{W}(z)$, $\mathbf{y}(k) = [y_1(k), \cdots, y_n(k)]^T$ is an $n$-dimensional vector of the outputs, which is used to estimate the source signals.

When we apply $\mathbf{W}(z)$ to the sensor signal $\mathbf{x}(k)$, the global transfer function from $\mathbf{s}(k)$ to $\mathbf{y}(k)$ is defined by $\mathbf{G}(z) = \mathbf{W}(z)\mathbf{H}(z)$. The goal of the blind deconvolution task is to find $\mathbf{W}(z)$ such that $\mathbf{G}(z) = \mathbf{P}\mathbf{\Lambda}\mathbf{D}(z)$, where $\mathbf{P} \in \mathbf{R}^{n\times n}$ is a permutation matrix, $\mathbf{D}(z) = \text{diag}\{z^{-d_1}, \cdots, z^{-d_n}\}$, and $\mathbf{\Lambda} \in \mathbf{R}^{n\times n}$ is a nonsingular diagonal scaling matrix.

## 3  Lie Group on $\mathcal{M}(N, N)$

In this section, we introduce a Lie group to the manifold of noncausal FIR filters. The Lie group operations play a crucial role in the following discussion. The set of all the noncausal FIR filters $\mathbf{W}(z)$ of length $N$, having the constraint that $\mathcal{W}$ is nonsingular, is denoted by

$$\mathcal{M}(N,N) = \left\{ \mathbf{W}(z) \mid \mathbf{W}(z) = \sum_{p=-N}^{N} \mathbf{W}_p z^{-p}, \, det(\mathcal{W}) \neq 0 \right\},\qquad(3)$$

where $\mathcal{W}$ is an $N \times N$ block matrix,

$$\mathcal{W} = \begin{bmatrix} \mathbf{W}_0 & \mathbf{W}_{-1} & \cdots & \mathbf{W}_{-N+1} \\ \mathbf{W}_1 & \mathbf{W}_0 & \cdots & \mathbf{W}_{-N+2} \\ \vdots & \vdots & \ddots & \vdots \\ \mathbf{W}_{N-1} & \mathbf{W}_{N-2} & \cdots & \mathbf{W}_0 \end{bmatrix}\qquad(4)$$

$\mathcal{M}(N, N)$ is a manifold of dimension $n^2(2N + 1)$. In general, multiplication of two filters in $\mathcal{M}(N, N)$ will enlarge the filter length and the result does belong to $\mathcal{M}(N, N)$ anymore. This makes it difficult to introduce the Riemannian structure to the manifold of noncausal FIR filters. In order to explore possible geometrical structures of $\mathcal{M}(N, N)$ which will lead to effective learning algorithms for $\mathbf{W}(z)$, we define algebraic operations of filters in the Lie group framework. First, we introduce a novel filter decomposition of noncausal filters in $\mathcal{M}(N, N)$ into a product of two one-sided FIR filters [19], which is illustrated in Fig. 1.

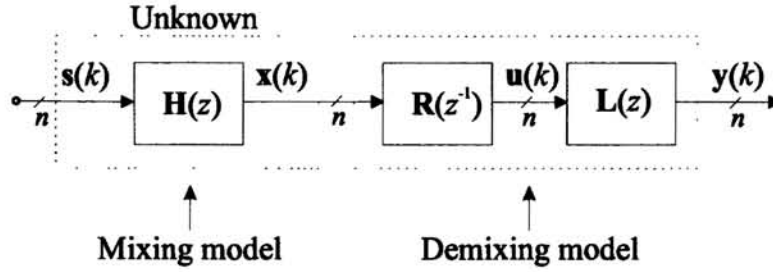

Figure 1: Illustration of decomposition of noncausal filters in $\mathcal{M}(N, N)$

**Lemma 1** *[19] If the matrix $\mathcal{W}$ is nonsingular, any noncausal filter $\mathbf{W}(z)$ in $\mathcal{M}(N,N)$ has the decomposition $\mathbf{W}(z) = \mathbf{R}(z)\mathbf{L}(z^{-1})$, where $\mathbf{R}(z) = \sum_{p=0}^{N} \mathbf{R}_p z^{-p}$, $\mathbf{L}(z^{-1}) = \sum_{p=0}^{N} \mathbf{L}_p z^p$ are one-sided FIR filters.*

In the manifold $\mathcal{M}(N, N)$, Lie operations, *multiplication* $*$ and *inverse* $\dagger$, are defined as follows: For $\mathbf{B}(z)$, $\mathbf{C}(z) \in \mathcal{M}(N, N)$,

$$\mathbf{B}(z) * \mathbf{C}(z) = [\mathbf{B}(z)\mathbf{C}(z)]_N, \qquad \mathbf{B}^\dagger(z) = \mathbf{L}^\dagger(z^{-1})\mathbf{R}^\dagger(z), \tag{5}$$

where $[\mathbf{B}(z)]_N$ is the truncating operator that any terms with orders higher than $N$ in the polynomial $\mathbf{B}(z)$ are truncated, and the inverse of one-side FIR filters is recurrently defined by $\mathbf{R}_0^\dagger = \mathbf{R}_0^{-1}$, $\mathbf{R}_p^\dagger = -\sum_{q=1}^{p} \mathbf{R}_{p-q}^\dagger \mathbf{R}_q \mathbf{R}_0^{-1}$, $p = 1, \cdots, N$. Refer to [18] for the detailed derivation. With these operations, both $\mathbf{B}(z) * \mathbf{C}(z)$ and $\mathbf{B}^\dagger(z)$ still remain in the manifold $\mathcal{M}(N, N)$. It is easy to verify that the manifold $\mathcal{M}(N, N)$ with the above operations forms a Lie Group. The identity element is $\mathbf{E}(z) = \mathbf{I}$.

# 4 Semiparametric Approach to Blind Deconvolution

We first introduce the basic theory of semiparametric models, and formulate blind deconvolution problem in the framework of the semiparametric models.

## 4.1 Semiparametric model

Consider a general statistical model $\{p(\mathbf{x}; \theta, \xi)\}$, where x is a random variable whose probability density function is specified by two parameters, $\theta$ and $\xi$, $\theta$ being the parameter of interest, and $\xi$ being the nuisance parameter. When the nuisance parameter is of infinite dimensions or of functional degrees of freedom, the statistical model is called a semiparametric model [12]. The gradient vectors of the log likelihood $\mathbf{u}(x; \theta, \xi) = \frac{\partial \log p(\mathbf{x}; \theta, \xi)}{\partial \theta}$, $\mathbf{v}(x; \theta, \xi) = \frac{\partial \log p(\mathbf{x}; \theta, \xi)}{\partial \xi}$, are called the score functions of the parameter of interest or shortly $\theta-$score and the nuisance score or shortly $\xi-$score, respectively.

In the semiparametric model, it is difficult to estimate both the parameters of interest and nuisance parameters at the same time, since the nuisance parameter $\xi$ is of infinite degrees of freedom. The semiparametric approach suggests to use an estimating function to estimate the parameters of interest, regardless of the nuisance parameters. The estimating function is a vector function $\mathbf{z}(\mathbf{x}, \theta)$, independent of nuisance parameters $\xi$, satisfying the following conditions

$$1) \quad E_{\theta, \xi}[\mathbf{z}(\mathbf{x}, \theta)] = 0, \tag{6}$$

$$2) \quad det(\mathcal{K}) \neq 0, \text{ where } \mathcal{K} = E_{\theta, \xi}[\frac{\partial \mathbf{z}(\mathbf{x}, \theta)}{\partial \theta}]. \tag{7}$$

$$3) \quad E_{\theta,\xi}[\mathbf{z}(\mathbf{x},\boldsymbol{\theta})\mathbf{z}^T(\mathbf{x},\boldsymbol{\theta})] < \infty, \tag{8}$$

for all $\boldsymbol{\theta}$ and $\boldsymbol{\xi}$. Generally speaking, it is difficult to find an estimating function. Amari and Kawanabe [9] studied the information geometry of estimating functions and provided a novel approach to find all the estimating functions. In this paper, we follow the approach to find a family of estimating functions for bind deconvolution.

## 4.2 Semiparametric Formulation for Blind Deconvolution

Now we turn to formulate the blind deconvolution problem in the framework of semiparametric models. From the statistical point of view, the blind deconvolution problem is to estimate $\mathbf{H}(z)$ or $\mathbf{H}^{-1}(z)$ from the observed data $\mathcal{D}_L = \{\mathbf{x}(k), k = 1, 2, \cdots\}$. The estimate includes two unknowns: One is the mixing filter $\mathbf{H}(z)$ which is the parameter of interest, and the other is the probability density function $p(\mathbf{s})$ of sources, which is the nuisance parameter in the present case. For blind deconvolution problem, we usually assume that source signals are zero-mean, $E[s_i] = 0$, for $i = 1, \cdots, n$. In addition, we generally impose constraints on the recovered signals to remove the indeterminacy,

$$E[k_i(s_i)] = 0, \text{ for } i = 1, \cdots, n. \tag{9}$$

A typical example of the constraint is $k_i(s_i) = s_i^4 - 1$. Since the source signals are spatially mutually independent and temporally iid, the pdf $r(\mathbf{s})$ can be factorized into a product form $r(\mathbf{s}) = \prod_{i=1}^{n} r(s_i)$. The purpose of this paper is to find a family of estimating functions for blind deconvolution. Remarkable progress has been made recently in the theory of the semiparametric approach [9],[12]. It has been shown that the efficient score itself is an estimating function for blind separation.

## 5  Estimating Functions

In this section, we give an explicit form of the score function matrix of interest and the nuisance tangent space, by using a local nonholonomic reparameterization. We then derive a family of estimating functions from it.

### 5.1  Score function matrix and its representation

Since the mixing model is a matrix filter, we write an estimating function in the same matrix filter format

$$\mathbf{F}(\mathbf{x}; \mathbf{H}(z)) = \sum_{p=-N}^{N} \mathbf{F}_p(\mathbf{x}; \mathbf{H}(z)) z^{-p}, \tag{10}$$

where $\mathbf{F}_p(\mathbf{x}; \mathbf{H}(z))$ are $n \times n$-matrices. In order to derive the explicit form of the $\mathbf{H}$-score, we reparameterize the filter in a small neighborhood of $\mathbf{H}(z)$ by using a new variable matrix filter as $\mathbf{H}(z) * (\mathbf{I} - \mathbf{X}(z))$, where $\mathbf{I}$ is the identity element of the manifold $\mathcal{M}(N, N)$. The variation $\mathbf{X}(z)$ represents a local coordinate system at the neighborhood $\mathcal{N}_{\mathbf{H}}$ of $\mathbf{H}(z)$ on the manifold $\mathcal{M}(N, N)$. The variation $d\mathbf{H}(z)$ of $\mathbf{H}(z)$ is represented as $d\mathbf{H}(z) = -\mathbf{H}(z) * d\mathbf{X}(z)$. Letting $\mathbf{W}(z) = \mathbf{H}^\dagger(z)$, we have

$$d\mathbf{X}(z) = d\mathbf{W}(z) * \mathbf{W}^\dagger(z), \tag{11}$$

which is a nonholonomic differential variable [6] since (11) is not integrable. With this representation of the parameters, we can obtain learning algorithms having the equivariant property [14] since the deviation $d\mathbf{X}(z)$ is independent of a specific $\mathbf{H}(z)$. The relative or the natural gradient of a cost function on the manifold can be automatically derived from this representation [2], [14], [18].

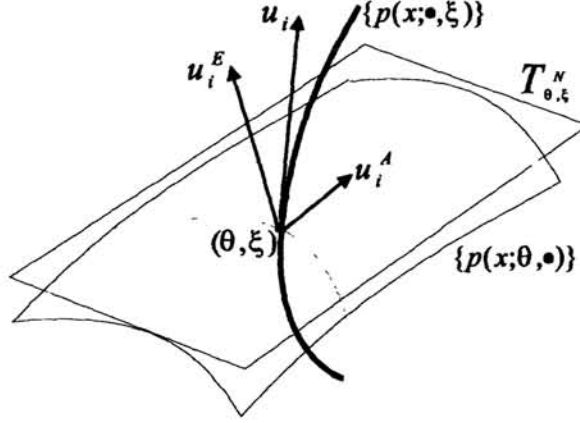

Figure 2: Illustration of orthogonal decomposition of score functions

The derivative of any cost function $l(\mathbf{H}(z))$ with respect to a noncausal filter $\mathbf{X}(z) = \sum_{p=-N}^{N} \mathbf{X}_p z^{-p}$ is defined by

$$\frac{\partial l(\mathbf{H}(z))}{\partial \mathbf{X}(z)} = \sum_{p=-N}^{N} \frac{\partial l(\mathbf{H}(z))}{\partial \mathbf{X}_p} z^{-p} \qquad (12)$$

Now we can easily calculate the score function matrix of noncausal filter $\mathbf{X}(z)$,

$$\frac{\partial \log p(\mathbf{x}; \mathbf{H}(z), r)}{\partial \mathbf{X}(z)} = \sum_{p=-N}^{N} \varphi(\mathbf{y}) \mathbf{y}^T (k - p) z^{-p}, \qquad (13)$$

where $\varphi(\mathbf{y}) = (\varphi_i(y_i), \cdots, \varphi_n(y_n))^T$, $\varphi_i(y_i) = \frac{d \log r_i(y_i)}{dy_i}$. and $\mathbf{y} = \mathbf{H}^\dagger(z)\mathbf{x}$.

## 5.2 Efficient scores

The efficient scores, denoted by $\mathbf{U}^E(\mathbf{s}; \mathbf{H}(z), r)$, can be obtained by projecting the score function to the space orthogonal to the nuisance tangent space $\mathcal{T}^N_{\mathbf{H}(z),r}$, which is illustrated in figure 2. In this section, we give an explicit form of the efficient scores for blind deconvolution.

**Lemma 2** *[5] The tangent nuisance space $\mathcal{T}^N_{\mathbf{H}(z),r}$ is a linear space spanned by the nuisance score functions, denoted by $\mathcal{T}^N_{\mathbf{H}(z),r} = \left\{ \sum_{i=1}^{n} c_i \alpha_i(s_i) \right\}$, where $c_i$ are coefficients, and $\alpha_i(s_i)$ are arbitrary functions, satisfying the following conditions*

$$E[\alpha_i(s_i)^2] < \infty, \quad E[s\alpha_i(s_i)] = 0, \quad E[k(s_i)\alpha_i(s_i)] = 0. \qquad (14)$$

We rewrite the score function (13) into the form $\mathbf{U}(\mathbf{s}; \mathbf{H}(z), r) = \sum_{p=-N}^{N} \mathbf{U}_p z^{-p}$, where $U_p = (\varphi(s_i(k))s_j(k-p))_{n \times n}$.

**Lemma 3** *The off-diagonal elements $u_{0,ij}(\mathbf{s}; \mathbf{H}(z), r)$, $i \neq j$, and the delay elements $u_{p,ij}(\mathbf{s}; \mathbf{H}(z), r)$, $p \neq 0$, of the score functions are orthogonal to the nuisance tangent space $\mathcal{T}^N_{\mathbf{H}(z),r}$.*

**Lemma 4** *The projection of $u_{0,ii}$ to the space orthogonal to the nuisance tangent space $\mathcal{T}^N_{\mathbf{H}(z),r}$ is of the form $w(s_i) = c_0 + c_1 s_i + c_2 k(s_i)$, where $c_i$ are any constants.*

In summary we have the following theorem

**Theorem 1** *The efficient score,* $\mathbf{U}^E(\mathbf{s}; \mathbf{H}(z), r) = \sum_{p=-N}^{N} \mathbf{U}_p^E z^{-p}$, *is given by*

$$\mathbf{U}_p^E = \varphi(\mathbf{s})\mathbf{s}^T(k-p), \text{ for } p \neq 0; \tag{15}$$

$$\mathbf{U}_0^E = \begin{cases} \varphi(\mathbf{s})\mathbf{s}^T, & \text{for off diagonal elements,} \\ c_0 + c_1 s + c_2 k(s), & \text{for diagonal elements.} \end{cases} \tag{16}$$

For the instantaneous mixture case, it has been proven [9] that the semiparametric model for blind separation is information m-curvature free. This is also true in the multichannel blind deconvolution case. As a result, the efficient score function is an estimating function for blind deconvolution. Using this result, we easily derive a family of estimating functions for blind deconvolution

$$\mathbf{F}(\mathbf{x}(k); \mathbf{W}(z)) = \sum_{p=-N}^{N} \varphi(\mathbf{y}(k))\mathbf{y}(k-p)^T z^{-p} - \mathbf{I}, \tag{17}$$

where $\mathbf{y}(k) = \mathbf{W}(z)\mathbf{x}(k)$, and $\varphi$ is a given function vector. The estimating function is the efficient score function, when $c_0 = c_1 = 0$, $c_2 = 1$ and $k_i(s_i) = \varphi_i(s_i)s_i - 1$.

## 6 Natural Gradient Learning and its Stability

Ordinary stochastic gradient methods for parameterized systems suffer from slow convergence due to the statistical correlations of the processes signals. While quasi-Newton and related methods can be used to improve convergence, they also suffer from the mass computation and numerical instability, as well as local convergence.

The natural gradient approach was developed to overcome the drawback of the ordinary gradient algorithm in the Riemannian spaces [2, 8, 15]. It has been proven that the natural gradient algorithm is an efficient algorithm in blind separation and blind deconvolution [2].

The efficient score function ( the estimating function ) gives an efficient search direction for updating filter $\mathbf{X}(z)$. Therefore, the updating rule for $\mathbf{X}(z)$ is described by

$$\mathbf{X}_{k+1}(z) = \mathbf{X}_k(z) - \eta \mathbf{F}(\mathbf{x}(k), \mathbf{W}_k(z)), \tag{18}$$

where $\eta$ is a learning rate. Since the new parameterization $\mathbf{X}(z)$ is defined by a nonholonomic transformation $d\mathbf{X}(z) = d\mathbf{W}(z) * \mathbf{W}^{\dagger}(z)$, the deviation of $\mathbf{W}(z)$ is given by

$$\Delta\mathbf{W}(z) = \Delta\mathbf{X}(z) * \mathbf{W}(z). \tag{19}$$

Hence, the natural gradient learning algorithm for $\mathbf{W}(z)$ is described as

$$\mathbf{W}_{k+1}(z) = \mathbf{W}_k(z) - \eta \mathbf{F}(\mathbf{x}(k), \mathbf{W}_k(z)) * \mathbf{W}_k(z), \tag{20}$$

where $\mathbf{F}(\mathbf{x}, \mathbf{W}(z))$ is an estimating function in the form (17). The stability of the algorithm (20) is equivalent to the one of algorithm (18). Consider the averaged version of algorithm (18)

$$\Delta\mathbf{X}(z) = -\eta E[\mathbf{F}(\mathbf{x}(k), \mathbf{W}_k(z))]. \tag{21}$$

Analyzing the variational equation of the above equation and using the mutual independence and i.i.d. properties of source signals, we derive the stability conditions of learning algorithm (21) at vicinity of the true solution

$$m_i + 1 > 0, \quad \kappa_i > 0, \quad \kappa_i \kappa_j \sigma_i^2 \sigma_j^2 > 1, \tag{22}$$

for $i, j = 1, \cdots, n$, where $m_i = E(\varphi'(y_i(k))y_i^2(k))$, $\kappa_i = E[\varphi_i'(y_i)]$, $\sigma_i^2 = E[|y_i|^2]$.

Therefore, we have the following theorem:

**Theorem 2** *If the conditions (22) are satisfied, then the natural gradient learning algorithm (20) is locally stable.*

# References

[1] S. Amari. *Differential–geometrical methods in statistics, Lecture Notes in Statistics,* volume 28. Springer, Berlin, 1985.

[2] S. Amari. Natural gradient works efficiently in learning. *Neural Computation,* 10:251–276, 1998.

[3] S. Amari. ICA of temporally correlated signals – Learning algorithm. In *Proceeding of 1st Inter. Workshop on Independent Component Analysis and Signal Separation,* pages 37–42, Aussois, France, January, 11-15 1999.

[4] S. Amari. Superefficiency in blind source separation. *IEEE Trans. on Signal Processing,* 47(4):936–944, April 1999.

[5] S. Amari and J.-F. Cardoso. Blind source separation– semiparametric statistical approach. *IEEE Trans. Signal Processing,* 45:2692–2700, Nov. 1997.

[6] S. Amari, T. Chen, and A. Cichocki. Nonholonomic orthogonal constraints in blind source separation. *Neural Comput.,* to be published.

[7] S. Amari and A. Cichocki. Adaptive blind signal processing– neural network approaches. *Proceedings of the IEEE,* 86(10):2026–2048, 1998.

[8] S. Amari, S. Douglas, A. Cichocki, and H. Yang. Multichannel blind deconvolution and equalization using the natural gradient. In *Proc. IEEE Workshop on Signal Processing Adv. in Wireless Communications,* pages 101–104, Paris, France, April 1997.

[9] S. Amari and M. Kawanabe. Estimating functions in semiparametric statistical models. In I. V. Basawa, V.P. Godambe, and R.L. Taylor, editors, *Estimating Functions,* volume 32 of *Monograph Series,* pages 65–81. IMS, 1998.

[10] S. Amari and M. Kumon. Estimation in the presence of infinitely many nuisance parameters in semiparametric statistical models. *Ann. Statistics,* 16:1044–1068, 1988.

[11] A.J. Bell and T.J. Sejnowski. An information maximization approach to blind separation and blind deconvolution. *Neural Computation,* 7:1129–1159, 1995.

[12] P. Bickel, C. Klaassen, Y. Ritov, and J. Wellner. *Efficient and Adaptive Estimation for Semiparametric Models.* The Johns Hopkins Univ. Press, Baltimore and London, 1993.

[13] J.-F Cardoso. Blind signal separation: Statistical principles. *Proceedings of the IEEE,* 86(10):2009–2025, 1998.

[14] J.-F. Cardoso and B. Laheld. Equivariant adaptive source separation. *IEEE Trans. Signal Processing,* SP-43:3017–3029, Dec 1996.

[15] A. Cichocki and R. Unbehauen. Robust neural networks with on-line learning for blind identification and blind separation of sources. *IEEE Trans Circuits and Systems I : Fundamentals Theory and Applications,* 43(11):894–906, 1996.

[16] L. Tong, R.W. Liu, V.C. Soon, and Y.F. Huang. Indeterminacy and identifiability of blind identification. *IEEE Trans. Circuits, Syst.,* 38(5):499–509, May 1991.

[17] H. Yang and S. Amari. Adaptive on-line learning algorithms for blind separation: Maximum entropy and minimal mutual information. *Neural Comput.,* 9:1457–1482, 1997.

[18] L. Zhang, A. Cichocki, and S. Amari. Geometrical structures of FIR manifold and their application to multichannel blind deconvolution. In *Proceeding of NNSP'99,* pages 303–312, Madison, Wisconsin, August 23-25 1999.

[19] L. Zhang, A. Cichocki, and S. Amari. Multichannel blind deconvolution of nonminimum phase systems using information backpropagation. In *Proceedings of the Fifth International Conference on Neural Information Processing(ICONIP'99),* page 210-216, Perth, Australia, Nov. 16-20 1999.
